# Adaptive Mixture of Probabilistic Transducers

**Yoram Singer**
AT&T Bell Laboratories
singer@research.att.com

## Abstract

We introduce and analyze a mixture model for supervised learning of probabilistic transducers. We devise an online learning algorithm that efficiently infers the structure and estimates the parameters of each model in the mixture. Theoretical analysis and comparative simulations indicate that the learning algorithm tracks the best model from an arbitrarily large (possibly infinite) pool of models. We also present an application of the model for inducing a noun phrase recognizer.

## 1 Introduction

Supervised learning of a probabilistic mapping between temporal sequences is an important goal of natural sequences analysis and classification with a broad range of applications such as handwriting and speech recognition, natural language processing and DNA analysis. Research efforts in supervised learning of probabilistic mappings have been almost exclusively focused on estimating the parameters of a *predefined* model. For example, in [5] a second order recurrent neural network was used to induce a finite state automata that classifies input sequences and in [1] an input-output HMM architecture was used for similar tasks.

In this paper we introduce and analyze an alternative approach based on a *mixture* model of a new subclass of probabilistic transducers, which we call suffix tree transducers. The mixture of experts architecture has been proved to be a powerful approach both theoretically and experimentally. See [4, 8, 6, 10, 2, 7] for analyses and applications of mixture models, from different perspectives such as connectionism, Bayesian inference and computational learning theory. By combining techniques used for compression [13] and unsupervised learning [12], we devise an online algorithm that efficiently updates the mixture weights *and* the parameters of *all* the possible models from an arbitrarily large (possibly infinite) pool of suffix tree transducers. Moreover, we employ the mixture estimation paradigm to the estimation of the parameters of each model in the pool and achieve an efficient estimate of the free parameters of each model. We present theoretical analysis, simulations and experiments with real data which show that the learning algorithm indeed tracks the best model in a *growing* pool of models, yielding an accurate approximation of the source. All proofs are omitted due to the lack of space

## 2 Mixture of Suffix Tree Transducers

Let $\Sigma_{in}$ and $\Sigma_{out}$ be two finite alphabets. A *Suffix Tree Transducer* $T$ over $(\Sigma_{in}, \Sigma_{out})$ is a rooted, $|\Sigma_{in}|$-ary tree where every internal node of $T$ has one child for each symbol in $\Sigma_{in}$. The nodes of the tree are labeled by pairs $(s, \gamma_s)$, where $s$ is the string associated with the path (sequence of symbols in $\Sigma_{in}$) that leads from the root to that node, and $\gamma_s : \Sigma_{out} \rightarrow [0, 1]$ is the output probability function. A suffix tree transducer (stochastically) maps arbitrarily long input sequences over $\Sigma_{in}$ to output sequences over $\Sigma_{out}$ as follows. The probability

that $T$ will output a string $y_1, y_2, \ldots, y_n$ in $\Sigma_{out}^n$ given an input string $x_1, x_2, \ldots, x_n$ in $\Sigma_{in}^n$, denoted by $P_T(y_1, y_2, \ldots, y_n | x_1, x_2, \ldots, x_n)$, is $\prod_{k=1}^n \gamma_{s^k}(y_k)$, where $s^1 = x_1$, and for $1 \le j \le n - 1$, $s^j$ is the string labeling the *deepest* node reached by taking the path corresponding to $x_j, x_{j-1}, x_{j-2}, \ldots$ starting at the root of $T$. A suffix tree transducer is therefore a probabilistic mapping that induces a measure over the possible output strings given an input string. Examples of suffix tree transducers are given in Fig. 1.

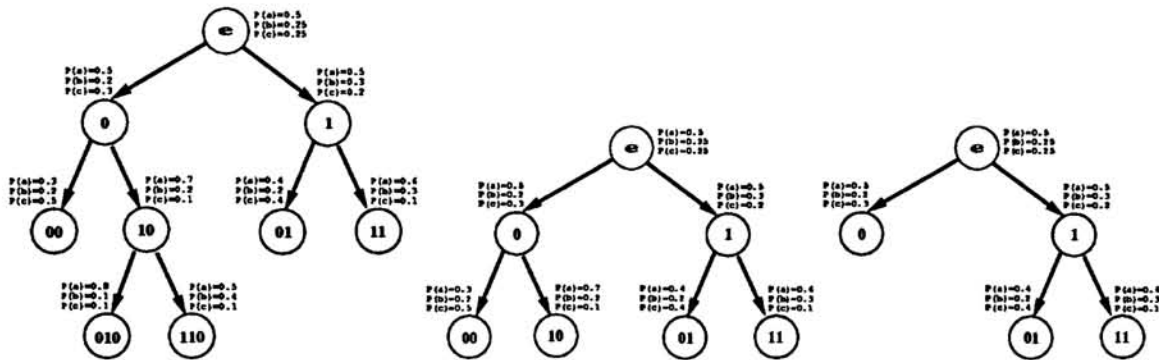

Figure 1: A suffix tree transducer (left) over $(\Sigma_{in}, \Sigma_{out}) = (\{0, 1\}, \{a, b, c\})$ and two of its possible sub-models (subtrees). The strings labeling the nodes are the suffixes of the input string used to predict the output string. At each node there is an output probability function defined for each of the possible output symbols. For instance, using the suffix tree transducer depicted on the left, the probability of observing the symbol b given that the input sequence is $\ldots, 0, 1, 0$, is 0.1. The probability of the current output, when each transducer is associated with a weight (prior), is the weighted sum of the predictions of each transducer. For example, assume that the weights of the trees are 0.7 (left tree), 0.2 (middle), and 0.1, then the probability that the output $y_n = a$ given that $(x_{n-2}, x_{n-1}, x_n) = (0, 1, 0)$ is $0.7 \cdot P_{T_1}(a|010) + 0.2 \cdot P_{T_2}(a|10) + 0.1 \cdot P_{T_3}(a|0) = 0.7 \cdot 0.8 + 0.2 \cdot 0.7 + 0.1 \cdot 0.5 = 0.75$.

Given a suffix tree transducer $T$ we are interested in the prediction of the mixture of *all* possible subtrees of $T$. We associate with each subtree (including $T$) a weight which can be interpreted as its prior probability. We later show how the learning algorithm of a mixture of suffix tree transducers adapts these weights with accordance to the performance (the *evidence* in Bayesian terms) of each subtree on past observations. Direct calculation of the mixture probability is infeasible since there might be exponentially many such subtrees. However, the technique introduced in [13] can be generalized and applied to our setting. Let $T'$ be a subtree of $T$. Denote by $n_1$ the number of the internal nodes of $T'$ and by $n_2$ the number of leaves of $T'$ which are *not* leaves of $T$. For example, $n_1 = 2$ and $n_2 = 1$, for the tree depicted on the right part of Fig. 1, assuming that $T$ is the tree depicted on the left part of the figure. The prior weight of a tree $T'$, denoted by $P_0(T')$ is defined to be $(1 - \alpha)^{n_1} \alpha^{n_2}$, where $\alpha \in (0, 1)$. Denote by $Sub(T)$ the set of all possible subtrees of $T$ including $T$ itself. It can be easily verified that this definition of the weights is a proper measure, i.e., $\sum_{T' \in Sub(T)} P_0(T') = 1$. This distribution over trees can be extended to unbounded trees assuming that the largest tree is an *infinite* $|\Sigma_{in}|$-ary suffix tree transducer and using the following randomized recursive process. We start with a suffix tree that includes only the root node. With probability $\alpha$ we stop the process and with probability $1 - \alpha$ we add all the possible $|\Sigma_{in}|$ sons of the node and continue the process recursively for each of the sons. Using this recursive prior the suffix tree transducers, we can calculate the prediction of the mixture at step $n$ in time that is linear in $n$, as follows,

$$\alpha \gamma_e(y_n) + (1 - \alpha)\left(\alpha \gamma_{x_n}(y_n) + (1 - \alpha)\left(\alpha \gamma_{x_{n-1}x_n}(y_n) + (1 - \alpha) \ldots\right.\right.$$

Therefore, the prediction time of a single symbol is bounded by the maximal depth of $T$, or the length of the input sequence if $T$ is infinite. Denote by $\tilde{\gamma}_s(y_n)$ the prediction of the mixture of subtrees rooted at $s$, and let $Leaves(T)$ be the set of leaves of $T$. The above

sum equals to $\bar{\gamma}_e(y_n)$, and can be evaluated recursively as follows,[1]

$$\bar{\gamma}_s(y_n) = \begin{cases} \gamma_s(y_n) & s \in Leaves(T) \\ \alpha\gamma_s(y_n) + (1-\alpha)\bar{\gamma}_{(x_{n-|s|},s)}(y_n) & \text{otherwise} \end{cases} \quad , \quad (1)$$

For example, given that the input sequence is $\ldots, 0, 1, 1, 0$, then the probabilities of the mixtures of subtrees for the tree depicted on the left part of Fig. 1, for $y_n = b$ and given that $\alpha = 1/2$, are, $\bar{\gamma}_{110}(b) = 0.4$ , $\bar{\gamma}_{10}(b) = 0.5 \cdot \gamma_{10}(b) + 0.5 \cdot 0.4 = 0.3$ , $\bar{\gamma}_0(b) = 0.5 \cdot \gamma_0(b) + 0.5 \cdot 0.3 = 0.25$ , $\bar{\gamma}_e(b) = 0.5 \cdot \gamma_e(b) + 0.5 \cdot 0.25 = 0.25$.

## 3  An Online Learning Algorithm

We now describe an efficient learning algorithm for a mixture of suffix tree transducers. The learning algorithm uses the recursive priors and the evidence to efficiently update the posterior weight of each possible subtree. In this section we assume that the output probability functions are known. Hence, we need to evaluate the following,

$$\begin{aligned} P(y_n|x_1,\ldots,x_n) &= \sum_{T' \in Sub(T)} P(y_n|T')P(T'|(x_1,y_1),\ldots,(x_{n-1},y_{n-1})) \\ &\overset{\text{def}}{=} \sum_{T' \in Sub(T)} P(y_n|T')P_n(T') \ , \end{aligned} \quad (2)$$

where $P_n(T')$ is the posterior weight of $T'$. Direct calculation of the above sum requires exponential time. However, using the idea of recursive calculation as in Equ. (1) we can efficiently calculate the prediction of the mixture. Similar to the definition of the recursive prior $\alpha$, we define $q_n(s)$ to be the *posterior* weight of a node $s$ compared to the mixture of all nodes below $s$. We can compute the prediction of the mixture of suffix tree transducers rooted at $s$ by simply replacing the prior weight $\alpha$ with the posterior weight, $q_{n-1}(s)$, as follows,

$$\bar{\gamma}_s(y_n) = \begin{cases} \gamma_s(y_n) & s \in Leaves(T) \\ q_{n-1}(s)\gamma_s(y_n) + (1 - q_{n-1}(s))\bar{\gamma}_{(x_{n-|s|},s)}(y_n) & \text{otherwise} \end{cases} \quad , \quad (3)$$

In order to update $q_n(s)$ we introduce one more variable, denoted by $r_n(s)$. Setting $r_0(s) = \log(\alpha/(1-\alpha))$ for all $s$, $r_n(s)$ is updated as follows,

$$r_n(s) = r_{n-1}(s) + \log(\gamma_s(y_n)) - \log(\bar{\gamma}_{x_{n-|s|}s}(y_n)) \ . \quad (4)$$

Therefore, $r_n(s)$ is the log-likelihood ratio between the prediction of $s$ and the prediction of the mixture of all nodes below $s$ in $T$. The new posterior weights $q_n(s)$ are calculated from $r_n(s)$,

$$q_n(s) = 1/(1 + e^{-r_n(s)}) \ . \quad (5)$$

In summary, for each new observation pair, we traverse the tree by following the path that corresponds to the input sequence $x_n x_{n-1} x_{n-2} \ldots$ The predictions of each sub-mixture are calculated using Equ. (3). Given these predictions the posterior weights of each sub-mixture are updated using Equ. (4) and Equ. (5). Finally, the probability of $y_n$ induced by the whole mixture is the prediction propagated out of the root node, as stated by Lemma 3.1.

**Lemma 3.1** $\sum_{T' \in Sub(T)} P(y_n|T')P_n(T') = \bar{\gamma}_e(y_n)$.

Let $Loss_n(T)$ be the logarithmic loss (negative log-likelihood) of a suffix tree transducer $T$ after $n$ input-output pairs. That is, $Loss_n(T) = \sum_{i=1}^{n} - \log(P(y_i|T))$. Similarly, the loss

of the mixture is defined to be, $Loss_n^{mix} = \sum_{i=1}^{n} -\log(\tilde{\gamma}_e(y_i))$. The advantage of using a mixture of suffix tree transducers over a single suffix tree is due to the robustness of the solution, in the sense that the prediction of the mixture is almost as good as the prediction of the best suffix tree in the mixture.

**Theorem 1** *Let $T$ be a (possibly infinite) suffix tree transducer, and let $(x_1, y_1), \ldots, (x_n, y_n)$ be any possible sequence of input-output pairs. The loss of the mixture is at most, $Loss_n(T') - \log(P_0(T'))$, for each possible subtree $T'$. The running time of the algorithm is $D\,n$ where $D$ is the maximal depth of $T$ or $n^2$ when $T$ is infinite.*

The proof is based on a technique introduced in [4]. Note that the additional loss is constant, hence the normalized loss per observation pair is, $P_0(T')/n$, which decreases like $O(\frac{1}{n})$.

Given a long sequence of input-output pairs or many short sequences, the structure of the suffix tree transducer is inferred as well. This is done by updating the output functions, as described in the next section, while *adding* new branches to the tree whenever the suffix of the input sequence does not appear in the current tree. The update of the weights, the parameters, and the structure ends when the maximal depth is reached, or when the beginning of the input sequence is encountered.

## 4 Parameter Estimation

In this section we describe how the output probability functions are estimated. Again, we devise an online scheme. Denote by $C_s^n(y)$ the number of times the output symbol $y$ was observed out of the $n$ times the node $s$ was visited. A commonly used estimator smoothes each count by adding a constant $\epsilon$ as follows,

$$\gamma_s(y) \approx \hat{\gamma}_s^n(y) \stackrel{def}{=} (C_s^n(y) + \epsilon)/(n + \epsilon\,|\Sigma_{out}|) \ . \tag{6}$$

The special case of $\epsilon = \frac{1}{2}$ is termed Laplace's modified rule of succession or the add$\frac{1}{2}$ estimator. In [9], Krichevsky and Trofimov proved that the loss of the add$\frac{1}{2}$ estimator, when applied sequentially, has a bounded logarithmic loss compared to the best (maximum-likelihood) estimator calculated *after* observing the entire input-output sequence. The additional loss of the estimator after $n$ observations is, $1/2(|\Sigma_{out}| - 1)\log(n) + |\Sigma_{out}| - 1$. When the output alphabet $\Sigma_{out}$ is rather small, we approximate $\gamma_s(y)$ by $\hat{\gamma}_s(y)$ using Equ. (6) and increment the count of the corresponding symbol every time the node $s$ is visited. We predict by replacing $\gamma$ with its estimate $\hat{\gamma}$ in Equ. (3). The loss of the mixture with estimated output probability functions, compared to any subtree $T'$ with *known* parameters, is now bounded as follows,

$$Loss_n^{mix} \leq Loss_n(T') - \log(P_0(T')) + 1/2\,|T'|\,(|\Sigma_{out}| - 1)\log(n/|T'|) + |T'|\,(|\Sigma_{out}| - 1) ,$$

where $|T'|$ is the number of leaves in $T'$. This bound is obtained by combining the bound on the prediction of the mixture from Thm. 1 with the loss of the smoothed estimator while applying Jensen's inequality [3].

When $|\Sigma_{out}|$ is fairly large or the sample size if fairly small, the smoothing of the output probabilities is too crude. However, in many real problems, only a small subset of the output alphabet is observed in a given context (a node in the tree). For example, when mapping *phonemes* to *phones* [11], for a given sequence of input phonemes the phones that can be pronounced is limited to a few possibilities. Therefore, we would like to devise an estimation scheme that statistically depends on the *effective* local alphabet and not on the whole alphabet. Such an estimation scheme can be devised by employing again a mixture of models, one model for each possible subset $\Sigma'_{out}$ of $\Sigma_{out}$. Although there are $2^{|\Sigma_{out}|}$ subsets of $\Sigma_{out}$, we next show that if the estimators depend only on the *size* of each subset then the whole mixture can be maintained in time *linear* in $|\Sigma_{out}|$.

Denote by $\hat{\gamma}_s^n(y\,|\,|\Sigma'_{out}| = i)$ the estimate of $\gamma_s(y)$ after $n$ observations given that the alphabet $\Sigma'_{out}$ is of size $i$. Using the add$\frac{1}{2}$ estimator, $\hat{\gamma}_s^n(y\,|\,|\Sigma'_{out}| = i) = (C_s^n(y) + 1/2)/(n + i/2)$. Let $\Sigma_{out}^n(s)$ be the set of different output symbols observed at node $s$, i.e.

$$\Sigma_{out}^n(s) = \left\{ \sigma \mid \sigma = y_{i_k}, \; s = (x_{i_k-|s|+1}, \dots, x_{i_k}), \; 1 \leq k \leq n \right\} \quad,$$

and define $\Sigma_{out}^0(s)$ to be the empty set. There are $\binom{|\Sigma_{out}| - |\Sigma_{out}^n(s)|}{i - |\Sigma_{out}^n(s)|}$ possible alphabets of size $i$. Thus, the prediction of the mixture of all possible subsets of $\Sigma_{out}$ is,

$$\hat{\gamma}_s^n(y) = \sum_{j=|\Sigma_{out}^n(s)|}^{|\Sigma_{out}|} \binom{|\Sigma_{out}| - |\Sigma_{out}^n(s)|}{j - |\Sigma_{out}^n(s)|} w_j^n \, \hat{\gamma}_s^n(y|j) \quad, \tag{7}$$

where $w_i^n$ is the posterior probability of an alphabet of size i. Evaluation of this sum requires $O(|\Sigma_{out}|)$ operations (and not $O(2^{|\Sigma_{out}|})$). We can compute Equ. (7) in an online fashion as follows. Let,

$$\hat{\Gamma}_s^n(i) \stackrel{\text{def}}{=} \sum_{\Sigma_{out}^n(s) \subseteq \Sigma'_{out},\, |\Sigma'_{out}|=i} w_i^0 \prod_{k=1}^n \hat{\gamma}_s^{k-1}(y_{i_k}\,|\Sigma'_{out})$$

$$= \binom{|\Sigma_{out}| - |\Sigma_{out}^n(s)|}{i - |\Sigma_{out}^n(s)|} w_i^0 \prod_{k=1}^n \hat{\gamma}_s^{k-1}(y_{i_k}|i) \quad. \tag{8}$$

Without loss of generality, let us assume a uniform prior for the possible alphabet sizes. Then,

$$P_0(\Sigma'_{out}) = P_0(|\Sigma'_{out}| = i) \stackrel{\text{def}}{=} w_i^0 = 1 / \left( |\Sigma_{out}| \binom{|\Sigma_{out}|}{i} \right) \quad.$$

Thus, for all $i$ $\hat{\Gamma}_s^0(i) = 1/|\Sigma_{out}|$. $\hat{\Gamma}_s^{n+1}(i)$ is updated from $\hat{\Gamma}_s^n(i)$ as follows,

$$\hat{\Gamma}_s^{n+1}(i) = \hat{\Gamma}_s^n(i) \times \begin{cases} 0 & \text{if } |\Sigma_{out}^{n+1}(s)| > i \\[6pt] \frac{C_s^n(y_{i_{n+1}})+1/2}{n+i/2} & \text{if } |\Sigma_{out}^{n+1}(s)| \leq i \text{ and } y_{i_{n+1}} \in \Sigma_{out}^n(s) \\[6pt] \frac{i - |\Sigma_{out}^n(s)|}{|\Sigma_{out}| - |\Sigma_{out}^n(s)|}\frac{1/2}{n+i/2} & \text{if } |\Sigma_{out}^{n+1}(s)| \leq i \text{ and } y_{i_{n+1}} \notin \Sigma_{out}^n(s) \end{cases}$$

Informally: If the number of different symbols observed so far exceeds a given size then all alphabets of this size are eliminated from the mixture by slashing their posterior probability to zero. Otherwise, if the next symbol was observed before, the output probability is the prediction of the $add\frac{1}{2}$ estimator. Lastly, if the next symbol is entirely new, we need to sum the predictions of all the alphabets of size $i$ which agree on the first $|\Sigma_{out}^n(s)|$ and $y_{i_{n+1}}$ is one of their $i - |\Sigma_{out}^n(s)|$ (yet) unobserved symbols. Furthermore, we need to multiply by the apriori probability of observing $y_{i_{n+1}}$. Assuming a uniform prior over the unobserved symbols this probability equals to $1/(|\Sigma_{out}| - |\Sigma_{out}^n(s)|)$. Applying Bayes rule again, the prediction of the mixture of all possible subsets of the output alphabet is,

$$\hat{\gamma}_s^n(y_{i_{n+1}}) = \sum_{i=1}^{|\Sigma_{out}|} \hat{\Gamma}_s^{n+1}(i) \Big/ \sum_{i=1}^{|\Sigma_{out}|} \hat{\Gamma}_s^n(i) \quad. \tag{9}$$

Applying twice the online mixture estimation technique, first for the structure and then for the parameters, yields an efficient and robust online algorithm. For a sample of size $n$, the time complexity of the algorithm is $D|\Sigma_{out}|n$ (or $|\Sigma_{out}|n^2$ if $\mathcal{T}$ is infinite). The predictions of the adaptive mixture is almost as good as *any* suffix tree transducer with *any* set of parameters. The logarithmic loss of the mixture depends on the number of *non-zero* parameters as follows,

$$Loss_n^{mix} \leq Loss_n(\mathcal{T}') - \log(P_0(\mathcal{T}')) + 1/2 \, l_{NZ} \log(n) + O(|\mathcal{T}'|\,|\Sigma_{out}|) \quad,$$

where $l_{NZ}$ is the number of non-zero parameters of the transducer $\mathcal{T}'$. If $l_{NZ} \ll |\mathcal{T}'|\,|\Sigma_{out}|$ then the performance of the above scheme, when employing a mixture model for the parameters as well, is significantly better than using the add$\frac{1}{2}$ rule with the full alphabet.

## 5   Evaluation and Applications

In this section we briefly present evaluation results of the model and its learning algorithm. We also discuss and present results obtained from learning syntactic structure of noun phrases. We start with an evaluation of the estimation scheme for a multinomial source.

In order to check the convergence of a mixture model for a multinomial source, we simulated a source whose output symbols belong to an alphabet of size 10 and set the probabilities of observing any of the last five symbols to zero. Therefore, the actual alphabet is of size 5. The posterior probabilities for the sum of all possible subsets of $\Sigma_{out}$ of size $i$ ($1 \leq i \leq 10$) were calculated after each iteration. The results are plotted on the left part of Fig. 2. The very first observations rule out alphabets of size lower than 5 by slashing their posterior probability to zero. After few observations, the posterior probability is concentrated around the actual size, yielding an accurate online estimate of the multinomial source.

The simplicity of the learning algorithm and the online update scheme enable evaluation of the algorithm on *millions* of input-output pairs in few minutes. For example, the average update time for a suffix tree transducer of a maximal depth 10 when the output alphabet is of size 4 is about 0.2 millisecond on a Silicon Graphics workstation. A typical result is shown in Fig. 2 on the right. In the example, $\Sigma_{out} = \Sigma_{in} = \{1, 2, 3, 4\}$. The description of the source is as follows. If $x_n \geq 3$ then $y_n$ is uniformly distributed over $\Sigma_{out}$, otherwise ($x_n \leq 2$) $y_n = x_{n-5}$ with probability 0.9 and $y_{n-5} = 4 - x_{n-5}$ with probability 0.1. The input sequence $x_1, x_2, \ldots$ was created entirely at random. This source can be implemented by a sparse suffix tree transducer of maximal depth 5. Note that the actual size of the alphabet is only 2 at half of the leaves of the tree. We used a suffix tree transducer of maximal depth 20 to learn the source. The negative of the logarithm of the predictions (normalized per symbol) are shown for (a) the true source, (b) a mixture of suffix tree transducers and their parameters, (c) a mixture of only the possible suffix tree transducers (the parameters are estimated using the $add\frac{1}{2}$ scheme), and (d) a single (overestimated) model of depth 8. Clearly, the mixture model converge to the entropy of the source much faster than the single model. Moreover, employing twice the mixture estimation technique results in an even faster convergence.

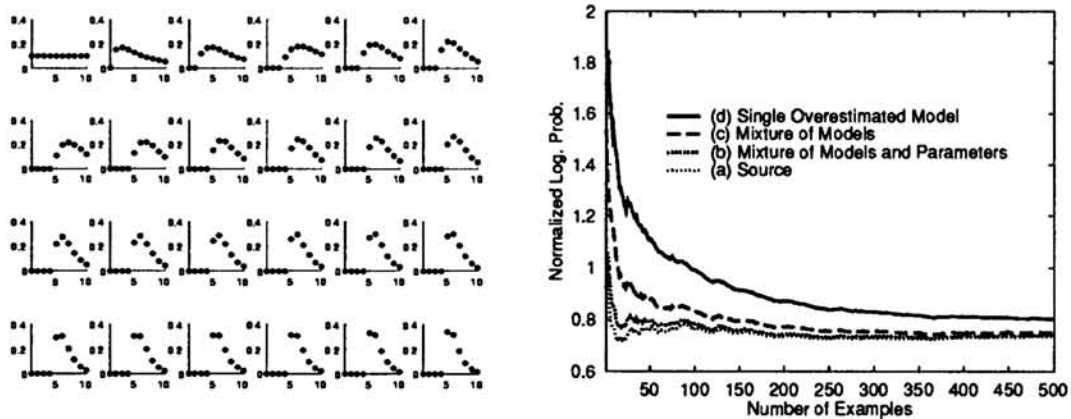

Figure 2: Left: Example of the convergence of the posterior probability of a mixture model for a multinomial source with large number of possible outcomes when the actual number of observed symbols is small. Right: performance comparison of the predictions of a single model, two mixture models and the true underlying transducer.

We are currently exploring the applicative possibilities of the algorithm. Here we briefly discuss and demonstrate how to induce an English noun phrase recognizer. Recognizing noun phrases is an important task in automatic natural text processing, for applications such as information retrieval, translation tools and data extraction from texts. A common practice is to recognize noun phrases by first analyzing the text with a part-of-speech tagger, which assigns the appropriate part-of-speech (verb, noun, adjective etc.) for each word in

context. Then, noun phrases are identified by manually defined regular expression patterns that are matched against the part-of-speech sequences. We took an alternative route by building a suffix tree transducer based on a labeled data set from the UPENN tree-bank corpus. We defined $\Sigma_{in}$ to be the set of possible part-of-speech tags and set $\Sigma_{out} = \{0, 1\}$, where the output symbol given its corresponding input symbol (the part-of-speech tag of the current word) is 1 *iff* the word is part of a noun phrase. We used over $250,000$ marked tags and tested the performance on more than $37,000$ tags. The test phase was performed by freezing the model structure, the mixture weights and the estimated parameters. The suffix tree transducer was of maximal depth 15 hence very long phrases can be statistically identified. By tresholding the output probability we classified the tags in the test data and found that less than 2.4% of the words were misclassified. A typical result is given in Table 1. We are currently investigating methods to incorporate linguistic knowledge into the model and its learning algorithm and compare the performance of the model with traditional techniques.

| Sentence | Tom | Smith | . | group | chief | executive | of | U.K. | metals |
|---|---|---|---|---|---|---|---|---|---|
| POS tag | PNP | PNP | . | NN | NN | NN | IN | PNP | NNS |
| Class | 1 | 1 | 0 | 1 | 1 | 1 | 0 | 1 | 1 |
| Prediction | 0.99 | 0.99 | 0.01 | 0.98 | 0.98 | 0.98 | 0.02 | 0.99 | 0.99 |
| Sentence | and | industrial | materials | maker | . | will | become | chairman | . |
| POS tag | CC | JJ | NNS | NN | . | MD | VB | NN | . |
| Class | 1 | 1 | 1 | 1 | 0 | 0 | 0 | 1 | 0 |
| Prediction | 0.67 | 0.96 | 0.99 | 0.96 | 0.03 | 0.03 | 0.01 | 0.87 | 0.01 |

Table 1: Extraction of noun phrases using a suffix tree transducer. In this typical example, two long noun phrases were identified correctly with high confidence.

## Acknowledgments

Thanks to Y. Bengio, Y. Freund, F. Pereira, D. Ron, R. Schapire, and N. Tishby for helpful discussions. The work on syntactic structure induction is done in collaboration with I. Dagan and S. Engelson. This work was done while the author was at the Hebrew University of Jerusalem.

## Footnotes

[1]A similar derivation still holds even if there is a different prior $\alpha_s$ at each node $s$ of $T$. For the sake of simplicity we assume that $\alpha$ is constant.

# References

[1] Y. Bengio and P. Fransconi. An input output HMM architecture. In *NIPS-7*, 1994.

[2] N. Cesa-Bianchi, Y. Freund, D. Haussler, D.P. Helmbold, R.E. Schapire, and M. K. Warmuth. How to use expert advice. In *STOC-24*, 1993.

[3] T.M. Cover and J.A. Thomas. *Elements of information theory*. Wiley, 1991.

[4] A. DeSantis, G. Markowski, and M.N. Wegman. Learning probabilistic prediction functions. In *Proc. of the 1st Wksp. on Comp. Learning Theory*, pages 312–328, 1988.

[5] C.L. Giles, C.B. Miller, D. Chen, G.Z. Sun, H.H. Chen, and Y.C. Lee. Learning and extracting finite state automata with second-order recurrent neural networks. *Neural Computation*, 4:393–405, 1992.

[6] D. Haussler and A. Barron. How well do Bayes methods work for on-line prediction of $\{+1, -1\}$ values ? In *The 3rd NEC Symp. on Comput. and Cogn.*, 1993.

[7] D.P. Helmbold and R.E. Schapire. Predicting nearly as well as the best pruning of a decision tree. In *COLT-8*, 1995.

[8] R.A. Jacobs, M.I. Jordan, S.J. Nowlan, and G.E. Hinton. Adaptive mixture of local experts. *Neural Computation*, 3:79–87, 1991.

[9] R.E. Krichevsky and V.K. Trofimov. The performance of universal encoding. *IEEE Trans. on Inform. Theory*, 1981.

[10] Nick Littlestone and Manfred K. Warmuth. The weighted majority algorithm. *Information and Computation*, 108:212–261, 1994.

[11] M.D. Riley. A statistical model for generating pronounication networks. In *Proc. of IEEE Conf. on Acoustics, Speech and Signal Processing*, pages 737–740, 1991.

[12] D. Ron, Y. Singer, and N. Tishby. The power of amnesia. In *NIPS-6*, 1993.

[13] F.M.J. Willems, Y.M. Shtarkov, and T.J. Tjalkens. The context tree weighting method: Basic properties. *IEEE Trans. Inform. Theory*, 41(3):653–664, 1995.